# Optimizing admission control while ensuring quality of service in multimedia networks via reinforcement learning*

**Timothy X Brown[†], Hui Tong[†], Satinder Singh[‡]**
† Electrical and Computer Engineering
‡ Computer Science
University of Colorado
Boulder, CO 80309-0425
{timxb, tongh, baveja}@colorado.edu

## Abstract

This paper examines the application of reinforcement learning to a telecommunications networking problem. The problem requires that revenue be maximized while simultaneously meeting a quality of service constraint that forbids entry into certain states. We present a general solution to this multi-criteria problem that is able to earn significantly higher revenues than alternatives.

## 1 Introduction

A number of researchers have recently explored the application of reinforcement learning (RL) to resource allocation and admission control problems in telecommunications. e.g., channel allocation in wireless systems, network routing, and admission control in telecommunication networks [1, 6, 7, 8]. Telecom problems are attractive applications for RL research because good, simple to implement, simulation models exist for them in the engineering literature that are both widely used and results on which are trusted, because there are existing solutions to compare with, because small improvements over existing methods can lead to significant savings in the long run, because they have discrete states, and because there are many potential commercial applications. However, existing RL applications have ignored an issue of great practical importance to telecom engineers, that of ensuring *quality of service* (QoS) while simultaneously optimizing whatever resource allocation performance criterion is of interest.

This paper will focus on admission control for broadband multimedia communication networks. These networks are unlike the current internet in that voice, video, and data calls arrive and depart over time and, in exchange for giving QoS guarantees to customers, the network collects revenue for calls that it accepts into the network. In this environment, admission control decides what calls to accept into the network so as to maximize the earned revenue while meeting the QoS guarantees of all carried customers.

Meeting QoS requires a decision function that decides when adding a new call will violate QoS guarantees. Given the diverse nature of voice, video, and data traffic, and their often complex underlying statistics, finding good QoS decision functions has been the subject of intense research [2, 5]. Recent results have emphasized that robust and efficient QoS decision functions require on-line adaptive methods [3].

Given we have a QoS decision function, deciding which of the heterogeneous arriving calls to accept and which to reject in order to maximize revenue can be framed as a dynamic program problem. The rapid growth in the number of states with problem complexity has led to reinforcement learning approaches to the problem [6].

In this paper we consider the problem of finding a control policy that simultaneously meets QoS guarantees and maximizes the network's earned revenue. We show that the straightforward approach of mixing positive rewards for revenue with negative rewards for violating QoS leads to sub-optimal policies. Ideally we would like to find the optimal policy from the subset of policies that never violate the QoS constraint. But there is no a priori useful way to characterize the space of policies that don't violate the QoS constraint. We present a general approach to meeting such multicriteria that solves this problem and potentially many other applications. Experiments show that incorporating QoS and RL yield significant gains over some alternative heuristics.

## 2 Problem Description

This section describes the admission control problem model that will be used. To emphasize the main features of the problem, networking issues such as queueing that are not essential have been simplified or eliminated. It should be emphasized that these aspects can readily be incorporated back into the problem.

We focus on a single network link. Users attempt to access the link over time and the network immediately chooses to accept or reject the call. If accepted, the call generates traffic in terms of bandwidth as a function of time. At a later time, the call terminates and departs from the network. For each call accepted, the network receives revenue at a fixed rate over the duration of the call. The network measures QoS metrics such as transmission delays or packet loss rates and compares them against the guarantees given to the calls. Thus, the problem is described by the call arrival, traffic, and departure processes; the revenue rates; QoS metrics; QoS constraints; and link model. The choices used in this paper are given in the next paragraph.

Calls are divided into discrete classes indexed by $i$. The calls are generated via a Poisson arrival process (arrival rate $\lambda_i$) and exponential holding times (mean holding time $1/\mu_i$). Within a call the bandwidth is an ON/OFF process where the traffic is either ON at rate $r_i$ or OFF at rate zero with mean holding times $\nu_i^{\text{ON}}$, and $\nu_i^{\text{OFF}}$. The effective immediate revenue are $c_i$. The link has a fixed bandwidth $B$. The total bandwidth used by accepted calls varies over time. The QoS metric is the fraction of time that the total bandwidth exceeds the link bandwidth (i.e. the overload probability, $p$). The QoS guarantee is an upper limit, $p^*$.

In previous work each call had a constant bandwidth over time so that the effect on QoS was predictable. Variable rate traffic is safely approximated by assuming that it always transmits at its maximum or peak rate. Such so-called *peak rate* allocation under-utilizes the network; in some cases by orders of magnitude less than what is possible. Stochastic traffic rates in real traffic, the desire for high network utilization/revenue, and the resulting potential for QoS violations distinguish the problem in this paper.

## 3 Semi-Markov Decision Processes

At any given point of time, the system is in a particular configuration, $x$, defined by the number of each type of ongoing calls. At random times a call arrival or a call termination

event, $e$, can occur. The configuration and event together determine the state of the system, $s = (x, e)$. When an event occurs, the learner has to choose an action feasible for that event. The choice of action, the event, and the configuration deterministically define the next configuration and the payoff received by the learner. Then after an interval the next event occurs, and this cycle repeats. The task of the learner is to determine a policy that maximizes the discounted sum of payoffs over an infinite horizon. Such a system constitutes a finite state, finite action, semi-Markov decision process (SMDP).

### 3.1  Multi-criteria Objective

The admission control objective is to learn a policy that assigns an accept or reject decision to each possible state of the system so as to *maximize*

$$J = E\left\{\int_0^\infty \gamma^t c(t)dt\right\},$$

where $E\{\cdot\}$ is the expectation operator, $c(t)$ is the total revenue rate of ongoing calls at time $t$, and $\gamma \in (0, 1)$ is a discount factor that makes immediate profit more valuable than future profit.[1]

In this paper we restrict the maximization to policies that never enter states that violate QoS guarantees. In general SMDP, due to stochastic state transitions, meeting such constraints may not be possible (e.g. from any state no matter what actions are taken there is a possibility of entering restricted states). In this problem service quality decreases with more calls in the system and adding calls is strictly controlled by the admission controller so that meeting this QoS constraint is possible.

### 3.2  Q-learning

RL methods solve SMDP problems by learning good approximations to the optimal value function, $J^*$, given by the solution to the Bellman optimality equation which takes the following form for the dynamic call admission problem:

$$J^*(s) \quad = \quad \max_{a \in A(s)} \left[ E_{\Delta t, s'} \{ c(s, a, \Delta t) + \gamma(\Delta t) J^*(s') \} \right] \qquad (1)$$

where $A(s)$ is the set of actions available in the current state $s$, $\Delta t$ is the random time until the next event, $c(s, a, \Delta t)$ is the effective *immediate* payoff with the discounting, and $\gamma(\Delta t)$ is the effective discount for the next state $s'$.

We learn an approximation to $J^*$ using Watkin's Q-learning algorithm. To focus on the dynamics of this paper's problem and not on the confounding dynamics of function approximation, the problem state space is kept small enough so that table lookup can be used. Bellman's equation can be rewritten in Q-values as

$$J^*(s) \quad = \quad \max_{a \in A(s)} Q^*(s, a) \qquad (2)$$

**Call Arrival:** When a call arrives, the Q-value of accepting the call and the Q-value of rejecting the call is determined. If rejection has the higher value, we drop the call. Else, if acceptance has the higher value, we accept the call.

**Call Termination:** No action needs to be taken.

Whatever our decision, we update our value function as follows: on a transition from state $s$ to $s'$ on action $a$ in time $\Delta t$,

$$Q(s, a) \quad = \quad (1 - \alpha)Q(s, a) + \alpha \left( c(s, a, \Delta t) + \gamma(\Delta t) \max_{b \in A(s')} Q(s', b) \right) \qquad (3)$$

where $\alpha \in [0,1]$ is the learning rate.

In order for Q-learning to perform well, all potentially important state-action pairs $(s,a)$ must be explored. At each state, with probability $\epsilon$ we apply an action that will lead to a less visited configuration, instead of the action recommended by the Q-value. However, to update Q-values we still use the action $b$ recommended by the Q-learning.

## 4 Combining Revenue and Quality of Service

The primary question addressed in this paper is how to combine the QoS constraint with the objective of maximizing revenue within this constraint. Let $\rho(s,a,\Delta t)$ and $q(s,a,\Delta t)$ be the revenue and measured QoS components of the reward, $c(s,a,\Delta t)$. Ideally $c(s,a,\Delta t) = \rho(s,a,\Delta t)$ when the QoS constraint is met and $c(s,a,\Delta t) = -\text{Large}$ (where $-\text{Large}$ is any large negative value) when QoS is not met. If the QoS parameters could be accurately measured between each state transition then this approach would be a valid solution to the problem. In network systems, the QoS metrics contain a high-degree of variability. For example, overload probabilities can be much smaller than $10^{-3}$ while the interarrival periods can be only a few ON/OFF cycles so that except for states with the most egregious QoS violations, most interarrival periods will have no overloads.

If the reward is a general function of revenue and QoS:

$$c(s,a,\Delta t) = f(\rho(s,a,\Delta t), q(s,a,\Delta t)), \qquad (4)$$

sufficient and necessary condition for inducing optimal policy with the QoS constraint is given by:

$$E\{f(\rho(s,a,\Delta t), q(s,a,\Delta t))\} = \begin{cases} E\{\rho(s,a,\Delta t)\} & \text{if } E\{q(s,a,\Delta t)\} < p^* \\ -\text{Large} & \text{otherwise} \end{cases} \qquad (5)$$

For $f(\cdot)$ satisfying this condition, states that violate QoS will be highly penalized and never visited. The actions for states that are visited will be based solely on revenue.

The Appendix gives a simple example showing that finding a $f(\cdot)$ that yields the optimal policy is unlikely without significant prior knowledge about each state. Several attempts at using (4) to combine QoS and revenue into the reward either violated QoS or had significantly lower reward.

A straight-forward alternative exists to meeting the multicriteria formulated as follows. For each criteria, $j$, we estimate a separate set of Q-factors, $Q^j(s,a)$. Each is updated via on-line Q-learning. These are then combined post facto at the time of decision via some function $\mathcal{Q}(\cdot)$ so that:

$$Q(s,a) = \mathcal{Q}(\{Q^j(s,a)\}). \qquad (6)$$

For example in this paper the two criteria are estimated separately as $Q^\rho$ and $Q^q$ and

$$Q(s,a) = \mathcal{Q}(Q^\rho(s,a), Q^q(s,a)) = \begin{cases} Q^\rho(s,a) & \text{if } Q^q(s,a) < p^* \\ -\text{Large} & \text{otherwise} \end{cases} \qquad (7)$$

The structure of this problem allows us to estimate $Q^q$ without using (3). As stated, the QoS is an intrinsic property of a state and not of future states so it is independent of the policy. This allows us to collect QoS statistics about each state and treat them in a principled way (e.g. computing confidence intervals on the estimates). Using these QoS estimates, the set of allowable states contracts monotonically over time eventually converging to a fixed set of allowable states. Since the QoS constraint is guaranteed to reach a fixed point asymptotically, the Q-learned policy also approaches a fixed point at the optimal policy via standard Q-learning proofs. A related scheme is analyzed in [4] suggesting that similar cases will also converge to optimal policies.

Many other QoS criteria do depend on the policy and require using (3). A constraint on the expected overload probability with a given policy is an example.

## 5   Simulation Results

The experiment uses the following model. The total bandwidth is normalized to 1.0 unit of traffic per unit time. The target overflow probability is $p^* = 10^{-3}$. Two source types are considered with the properties shown in Table 1. As noted before, call holding times are exponential and the arrivals are Poisson. For the first experiment, the ON/OFF holding times are exponentially distributed, while for the second experiment, they are Pareto distributed. The Pareto distribution is considered to be a more accurate representation of data traffic.

Table 1: Experimental parameters

|  | Source Type | |
|---|---|---|
| *Parameter* | *I* | *II* |
| ON rate, $r$ | 0.08 | 0.2 |
| Mean ON period, $1/\nu^{ON}$ | 5 | 5 |
| Mean OFF period, $1/\nu^{OFF}$ | 15 | 45 |
| Hyperbolic exponent, $u + 1$ | 2.08 | 2.12 |
| Call arrival rate, $\lambda$ | 0.067 | 0.2 |
| Call holding time, $1/\mu$ | 60 | 60 |
| Immediate payoff, $c$ | 5 | 1 |

In the experiments, for each state-action pair, $(s, a)$, $Q^p(s, a)$ is updated using (3). As stated, in this case the update of $Q^q(s, a)$ does not need to use (3). Since random exploration is employed to ensure that all potentially important state-action pairs be tried, it naturally enables us to collect statistics that can be used to estimate QoS at these state-action pairs, $Q^q(s, a)$. As the number of visits to each state-action pair increases, the estimated $Q^q(s, a)$ becomes more and more accurate and, with confidence, we can gradually eliminate those state-action pairs that will violate QoS requirement. As a consequence, $Q^p(s, a)$ is updated in a gradually correct subset of state-action space in the sense that QoS is met for any action within this subspace. Initial Q-values for RL are artificially set such that Q-learning started with the *greedy* policy (the greedy policy always accepts).

After training is completed, we apply a test data set to compare the policy obtained through RL with alternative heuristic policies. The final QoS measurements obtained at the end of the RL training while learning QoS are used for testing different policies. To test the RL policies, when there is a new call arrival, the algorithm first determines if accepting this call will violate QoS. If it will, the call is rejected, else the action is chosen according to $a = \arg\max_{a \in A(s)} Q(s, a)$, where $A(s) = \{1=\text{accept}, 0=\text{reject}\}$. For the QoS constraint we use three cases: Peak rate allocation; Statistical multiplexing function learned on-line, denoted QoS learned; Given statistical multiplexing function a priori, denoted QoS given. We examine six different cases: (1) RL: QoS given; (2) RL: QoS learned; (3) RL: peak rate; (4) A heuristic that only accepts calls from the most valuable class, i.e., type I, with given QoS; (5) Greedy: QoS given; (6) Greedy: peak rate.

From the results shown in Fig. 1, it is clear that simultaneously doing Q-learning and QoS learning converges correctly to the RL policy obtained by giving the QoS a priori and doing standard Q-learning only. We see significant gains (about 15%) due to statistical multiplexing: (6) vs (5), and (3) vs (1). The gains due to RL are about 25%: (6) vs (3), and (5) vs (2). Together they yield about 45% increase in revenue over conservative peak rate allocation in this example. It is also clear from the figure that the RL policies perform better than the heuristic policies. Fig. (2) shows the rejection ratios for different policies.

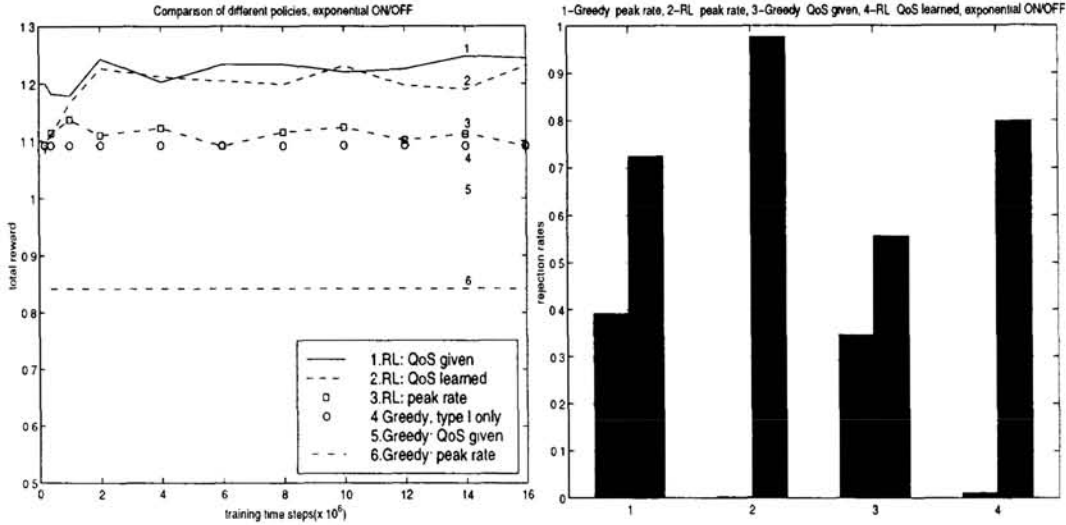

Figure 1: Comparison of total rewards of RL while learning QoS, RL with given QoS measurements, RL with peak rate, greedy policies and peak rate allocation, normalized by the greedy total reward – exponential ON/OFF.

Figure 2: Comparison of rejection ratios for the policies learned in Fig. 1.

We repeat the above experiments with Pareto distributed ON and OFF periods, using the same parameters listed in Table 1. The results are shown in Figs. 3–4. Clearly, the different ON/OFF distributions yield similar gains for RL.

## 6    Conclusion

This paper shows that a QoS constraint could be incorporated into a RL solution to maximizing a network's revenue, using a vector value Q-learning function. The formulation is quite general and can be applied to many possible constraints. The approach, when applied to a simple networking problem, increases revenue by up to 45%. Future research includes: using neural networks or other function approximators to deal with more complex problems for which lookup tables are infeasible; and extending admission control to multi-link routing.

## 7    Appendix: Simple One-State Example

A simple example will show that a function with property (5) is unlikely. Consider a link that can accept only one type of call and it can accept no more than one call. With no actions possible when carrying a call there is only one state. Only two rewards are possible, $c(R)$ for reject and $c(A)$ for accept. To fix the value function let $c(R) = 0$ and let $\rho$ and $q$ be the random revenues and QoS experienced. Analysis of (1) and (2) shows that the accept action will be chosen if and only if $E\{f(\rho, q)\} > 0$.

In this example, the revenues are random and possibly negative (e.g. if they are net after cost of billing and transport). The call should be accepted if $E\{\rho\} > 0$ and $E\{q\} < p^*$. Therefore the correct reward function has the property:

$$E\{f(\rho, q)\} > 0 \qquad \text{if } E\{\rho\} > 0 \text{ and } E\{q\} < p^* \tag{8}$$

The point of the example is that an $f(\cdot)$ satisfying (8) requires prior knowledge about the distributions of the revenue and the QoS as a function of the state. Even if it were possible

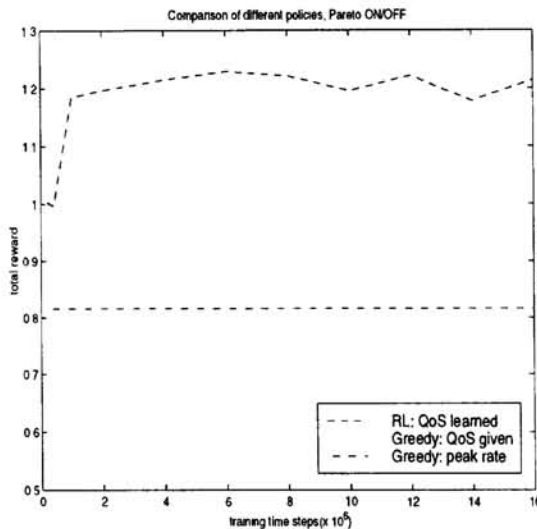
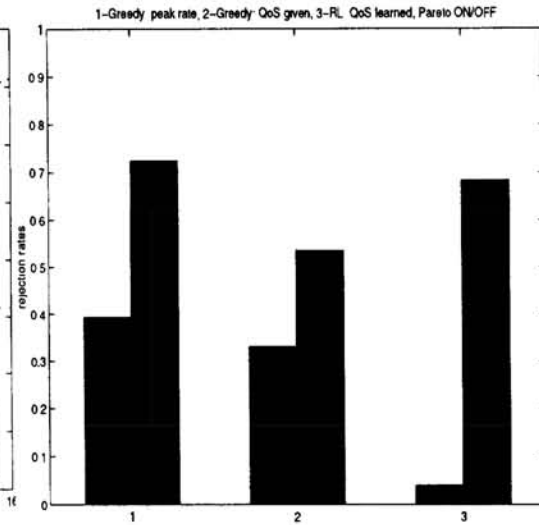

Figure 3: Comparison of total rewards of RL while learning QoS, greedy policy and peak rate allocation, normalized by the greedy total reward – Pareto ON/OFF.

Figure 4: Comparison of rejection ratios for the policies learned in Fig. 3.

for this example, setting up constraints such as (8) for a real problem with a huge state space would be non-trivial because $\rho$ and $q$ are functions of the many state and action pairs.

## Footnotes

*Timothy Brown and Hui Tong were funded by NSF CAREER Award NCR-9624791. Satinder Singh was funded by NSF grant IIS-9711753.

[1] Since we will compare policies based on total reward rather than discounted sum of reward, we can use the Tauberian approximation [4], i.e., $\gamma$ is chosen to be sufficiently close to 1.

## References

[1] Boyan, J.A., Littman, M.L., "Packet routing in dynamically changing networks: a reinforcement learning approach," in Cowan, J.D., et al., ed. *Advances in NIPS 6*, Morgan Kauffman, SF, 1994. pp. 671–678.

[2] Brown, T.X, "Adaptive Access Control Applied to Ethernet Data," *Advances in NIPS 9*, ed. M. Mozer et al., MIT Press, 1997. pp. 932–938.

[3] Brown, T.X, "Adaptive Statistical Multiplexing for Broadband Communications," Invited Tutorial *Fifth IFIP Workshop on Performance Modeling & Evaluation of ATM Networks*, Ilkley, U.K., July, 1997.

[4] Gabor, Z., Kalmar, Z., Szepesvari, C., "Multi-criteria Reinforcement Learning," to appear in *International Conference on Machine Learning*, Madison, WI, July, 1998.

[5] Hiramatsu, A., "ATM Communications Network Control by Neural Networks," *IEEE T. on Neural Networks*, v. 1, n. 1, pp. 122–130, 1990.

[6] Marbach, P., Mihatsch, O., Schulte, M., Tsitsiklis, J.N., "Reinforcement learning for call admission control and routing in integrated service networks," in Jordan, M., et al., ed. *Advances in NIPS 10*, MIT Press, 1998.

[7] Nie, J., Haykin, S., "A Q-learning based dynamic channel assignment technique for mobile communication systems," to appear in *IEEE T. on Vehicular Technology*.

[8] Singh, S.P., Bertsekas, D.P., "Reinforcement learning for dynamic channel allocation in cellular telephone systems," in *Advances in NIPS 9*, ed. Mozer, M., et al., MIT Press, 1997. pp. 974–980.
